# Optimal Scoring for Unsupervised Learning

**Zhihua Zhang** and **Guang Dai**
College of Computer Science & Technology
Zhejiang University
Hangzhou, Zhejiang, 310027 China

## Abstract

We are often interested in casting classification and clustering problems as a regression framework, because it is feasible to achieve some statistical properties in this framework by imposing some penalty criteria. In this paper we illustrate optimal scoring, which was originally proposed for performing the Fisher linear discriminant analysis by regression, in the application of unsupervised learning. In particular, we devise a novel clustering algorithm that we call *optimal discriminant clustering*. We associate our algorithm with the existing unsupervised learning algorithms such as spectral clustering, discriminative clustering and sparse principal component analysis. Experimental results on a collection of benchmark datasets validate the effectiveness of the optimal discriminant clustering algorithm.

## 1 Introduction

The Fisher linear discriminant analysis (LDA) is a classical method that considers dimensionality reduction and classification jointly. LDA estimates a low-dimensional discriminative space defined by linear transformations through maximizing the ratio of between-class scatter to within-class scatter. It is well known that LDA is equivalent to a least mean squared error procedure in the binary classification problem [4]. It is of great interest to obtain a similar relationship in multi-class problems. A significant literature has emerged to address this issue [6, 8, 12, 14]. This provides another approach to performing LDA by regression, in which penalty criteria are tractably introduced to achieve some statistical properties such as regularized LDA [5] and sparse discriminant analysis [2].

It is also desirable to explore unsupervised learning problems in a regression framework. Recently, Zou *et al.* [17] reformulated principal component analysis (PCA) as a regression problem and then devised a sparse PCA by imposing the lasso (the elastic net) penalty [10, 16] on the regression vector. In this paper we consider unsupervised learning problems by optimal scoring, which was originally proposed to perform LDA by regression [6]. In particular, we devise a novel unsupervised framework by using the optimal scoring and the ridge penalty.

This framework can be used for dimensionality reduction and clustering simultaneously. We are mainly concerned with the application in clustering. In particular, we propose a clustering algorithm that we called *optimal discriminant clustering* (ODC). Moreover, we establish a connection of our clustering algorithm with discriminative clustering algorithms [3, 13] and spectral clustering algorithms [7, 15]. This implies that we can cast these clustering algorithms as regression-type problems. In turn, this facilitates the introduction of penalty terms such as the lasso and elastic net so that we have sparse unsupervised learning algorithms.

Throughout this paper, $\mathbf{I}_m$ denotes the $m \times m$ identity matrix, $\mathbf{1}_m$ the $m \times 1$ vector of ones, $\mathbf{0}$ the zero vector or matrix with appropriate size, and $\mathbf{H}_m = \mathbf{I}_m - \frac{1}{m}\mathbf{1}_m\mathbf{1}_m'$ the $m \times m$ centering matrix. For an $m \times 1$ vector $\mathbf{a} = (a_1, \ldots, a_m)'$, $\mathrm{diag}(\mathbf{a})$ represents the $m \times m$ diagonal matrix with $a_1, \ldots, a_m$ as its diagonal entries. For an $m \times m$ matrix $\mathbf{A} = [a_{ij}]$, we let $\mathbf{A}^+$ be the Moore-Penrose inverse of $\mathbf{A}$, $\mathrm{tr}(\mathbf{A})$ be the trace of $\mathbf{A}$, $\mathrm{rk}(\mathbf{A})$ be the rank of $\mathbf{A}$ and $\|\mathbf{A}\|_F = \sqrt{\mathrm{tr}(\mathbf{A}'\mathbf{A})}$ be the Frobenius norm of $\mathbf{A}$.

## 2 Problem Formulation

We are concerned with a multi-class classification problem. Given a set of $n$ $p$-dimensional data points, $\{\mathbf{x}_1, \ldots, \mathbf{x}_n\} \in \mathcal{X} \subset \mathbb{R}^p$, we assume that the $\mathbf{x}_i$ are grouped into $c$ disjoint classes and that each $\mathbf{x}_i$ belongs to one class. Let $V = \{1, 2, \ldots, n\}$ denote the index set of the data points $\mathbf{x}_i$ and partition $V$ into $c$ disjoint subsets $V_j$; i.e., $V_i \cap V_j = \varnothing$ for $i \neq j$ and $\cup_{j=1}^c V_j = V$, where the cardinality of $V_j$ is $n_j$ so that $\sum_{j=1}^c n_j = n$.

We also make use of a matrix representation for the problem in question. In particular, we let $\mathbf{X} = [\mathbf{x}_1, \ldots, \mathbf{x}_n]'$ be an $n \times p$ data matrix, and $\mathbf{E} = [e_{ij}]$ be an $n \times c$ indicator matrix with $e_{ij} = 1$ if input $\mathbf{x}_i$ is in class $j$ and $e_{ij} = 0$ otherwise. Let $\mathbf{\Pi} = \mathrm{diag}(n_1, \ldots, n_c)$, $\mathbf{\Pi}^{\frac{1}{2}} = \mathrm{diag}(\sqrt{n_1}, \ldots, \sqrt{n_c})$, $\boldsymbol{\pi} = (n_1, \ldots, n_c)'$ and $\sqrt{\boldsymbol{\pi}} = (\sqrt{n_1}, \ldots, \sqrt{n_c})'$. It follows that $\mathbf{1}_n' \mathbf{E} = \mathbf{1}_c' \mathbf{\Pi} = \boldsymbol{\pi}'$, $\mathbf{E}\mathbf{1}_c = \mathbf{1}_n$, $\mathbf{1}_c' \boldsymbol{\pi} = n$, $\mathbf{E}'\mathbf{E} = \mathbf{\Pi}$ and $\mathbf{\Pi}^{-1} \boldsymbol{\pi} = \mathbf{1}_c$.

### 2.1 Scoring Matrices

Hastie *et al.* [6] defined a scoring matrix for the $c$-class classification problem. That is, it is such a $c \times (c-1)$ matrix $\mathbf{\Theta} \in \mathbb{R}^{c \times (c-1)}$ that $\mathbf{\Theta}'(\mathbf{E}'\mathbf{E})\mathbf{\Theta} = \mathbf{\Theta}'\mathbf{\Pi}\mathbf{\Theta} = \mathbf{I}_{c-1}$. The $j$th row of $\mathbf{\Theta}$ defines a *scoring* or *scaling* for the $j$th class. Here we refine this definition as:

**Definition 1** *Given a c-class classification problem with the cardinality of the jth class being $n_j$, a $c \times (c-1)$ matrix $\mathbf{\Theta}$ is referred to as the* class scoring matrix *if it satisfies*

$$\mathbf{\Theta}'\mathbf{\Pi}\mathbf{\Theta} = \mathbf{I}_{c-1} \quad and \quad \boldsymbol{\pi}'\mathbf{\Theta} = \mathbf{0}.$$

It follows from this definition that $\mathbf{\Theta}\mathbf{\Theta}' = \mathbf{\Pi}^{-1} - \frac{1}{n}\mathbf{1}_c\mathbf{1}_c'$. In the literature [15], the authors presented a specific example for $\mathbf{\Theta} = (\boldsymbol{\theta}_1, \ldots, \boldsymbol{\theta}_{c-1})'$. That is, $\boldsymbol{\theta}_1' = \left( \frac{\sqrt{n-n_1}}{\sqrt{nn_1}}, -\frac{\sqrt{n_1}}{\sqrt{n(n-n_1)}}\mathbf{1}_{c-1}' \right)$ and

$$\boldsymbol{\theta}_l' = \left( 0 * \mathbf{1}_{l-1}', \frac{\sqrt{\sum_{j=l+1}^c n_j}}{\sqrt{n_l \sum_{j=l}^c n_j}}, \frac{\sqrt{n_l}}{\sqrt{\sum_{j=l}^c n_j \sum_{j=l+1}^c n_j}}\mathbf{1}_{c-l}' \right)$$

for $l = 2, \ldots, c-1$. Especially, when $c = 2$, $\mathbf{\Theta} = (\frac{\sqrt{n_2}}{\sqrt{nn_1}}, -\frac{\sqrt{n_1}}{\sqrt{nn_2}})'$ is a 2-dimensional vector.

Let $\mathbf{Y} = \mathbf{E}\mathbf{\Theta}$ ($n \times (c-1)$). We then have $\mathbf{Y}'\mathbf{Y} = \mathbf{I}_{c-1}$ and $\mathbf{1}_n'\mathbf{Y} = \mathbf{0}$. To address an unsupervised clustering problem with $c$ classes, we relax the setting of $\mathbf{Y} = \mathbf{E}\mathbf{\Theta}$ and give the following definition.

**Definition 2** *An $n \times (c-1)$ matrix $\mathbf{Y}$ is referred to as the* sample scoring matrix *if it satisfies*

$$\mathbf{Y}'\mathbf{Y} = \mathbf{I}_{c-1} \quad and \quad \mathbf{1}_n'\mathbf{Y} = \mathbf{0}.$$

Note that $c$ does not necessarily represent the number of classes in this definition. For example, we view $c-1$ as the dimension of a reduced dimensional space in the dimensionality reduction problem.

### 2.2 Optimal Scoring for LDA

To devise a classifier for the $c$-class classification problem, we consider a penalized optimal scoring model, which is defined by

$$\min_{\mathbf{\Theta}, \mathbf{W}} \left\{ f(\mathbf{\Theta}, \mathbf{W}) \triangleq \frac{1}{2} \|\mathbf{E}\mathbf{\Theta} - \mathbf{H}_n\mathbf{X}\mathbf{W}\|_F^2 + \frac{\sigma^2}{2}\mathrm{tr}(\mathbf{W}'\mathbf{W}) \right\} \tag{1}$$

under the constraints $\mathbf{\Theta}'\mathbf{\Pi}\mathbf{\Theta} = \mathbf{I}_{c-1}$ and $\boldsymbol{\pi}'\mathbf{\Theta} = \mathbf{0}$ where $\mathbf{\Theta} \in \mathbb{R}^{c \times (c-1)}$ and $\mathbf{W} \in \mathbb{R}^{p \times (c-1)}$. Compared with the setting in [6], we add the constraint $\boldsymbol{\pi}'\mathbf{\Theta} = \mathbf{0}$. The reason is due to $\mathbf{1}_n'\mathbf{H}_n\mathbf{X}\mathbf{W} = \mathbf{0}$. We thus impose $\mathbf{1}_n'\mathbf{E}\mathbf{\Theta} = \boldsymbol{\pi}'\mathbf{\Theta} = \mathbf{0}$ for consistency.

Denote

$$\mathbf{R} = \mathbf{\Pi}^{-\frac{1}{2}}\mathbf{E}'\mathbf{H}_n\mathbf{X}(\mathbf{X}'\mathbf{H}_n\mathbf{X} + \sigma^2\mathbf{I}_p)^{-1}\mathbf{X}'\mathbf{H}_n\mathbf{E}\mathbf{\Pi}^{-\frac{1}{2}}.$$

Since $\mathbf{R}\boldsymbol{\pi}^{\frac{1}{2}} = \mathbf{0}$, there exists a $c \times (c-1)$ orthogonal matrix $\mathbf{\Delta}$, the columns of which are the eigenvectors of $\mathbf{R}$. That is, $\mathbf{\Delta}$ satisfies $\mathbf{\Delta}'\mathbf{\Delta} = \mathbf{I}_{c-1}$ and $\mathbf{\Delta}'\boldsymbol{\pi}^{\frac{1}{2}} = \mathbf{0}$.

**Theorem 1** *A minimizer of Problem (1) is* $\hat{\boldsymbol{\Theta}} = \boldsymbol{\Pi}^{-\frac{1}{2}}\boldsymbol{\Delta}$ *and* $\hat{\mathbf{W}} = (\mathbf{X}'\mathbf{H}_n\mathbf{X} + \sigma^2\mathbf{I}_p)^{-1}\mathbf{X}'\mathbf{H}_n\mathbf{E}\hat{\boldsymbol{\Theta}}$. *Here* $[\boldsymbol{\Delta}, \frac{1}{\sqrt{n}}\boldsymbol{\pi}^{\frac{1}{2}}]$ *is the* $c \times c$ *matrix of the orthonormal eigenvectors of* $\mathbf{R}$.

Since for an arbitrary class scoring matrix $\boldsymbol{\Theta}$, its rank is $c-1$, we have $\boldsymbol{\Theta} = \hat{\boldsymbol{\Theta}}\boldsymbol{\Upsilon}$ where $\boldsymbol{\Upsilon}$ is some $(c-1) \times (c-1)$ orthonormal matrix. Moreover, it follows from $\boldsymbol{\Theta}\boldsymbol{\Theta}' = \boldsymbol{\Pi}^{-1} - \frac{1}{n}\mathbf{1}_c\mathbf{1}_c'$ that the between-class scatter matrix is given by

$$\boldsymbol{\Sigma}_b = \mathbf{X}'\mathbf{H}_n\mathbf{E}\boldsymbol{\Theta}\boldsymbol{\Theta}'\mathbf{E}'\mathbf{H}_n\mathbf{X} = \mathbf{X}'\mathbf{H}_n\mathbf{E}\hat{\boldsymbol{\Theta}}\hat{\boldsymbol{\Theta}}'\mathbf{E}'\mathbf{H}_n\mathbf{X}.$$

Accordingly, we can also write the generalized eigenproblem for the penalized LDA as

$$\mathbf{X}'\mathbf{H}_n\mathbf{E}\hat{\boldsymbol{\Theta}}\hat{\boldsymbol{\Theta}}'\mathbf{E}'\mathbf{H}_n\mathbf{X}\mathbf{A} = (\mathbf{X}'\mathbf{H}_n\mathbf{X} + \sigma^2\mathbf{I}_p)\mathbf{A}\boldsymbol{\Lambda},$$

because the total scatter matrix $\boldsymbol{\Sigma}$ is $\boldsymbol{\Sigma} = \mathbf{X}'\mathbf{H}_n\mathbf{X}$. We now obtain

$$\hat{\mathbf{W}}\hat{\boldsymbol{\Theta}}'\mathbf{E}'\mathbf{H}_n\mathbf{X}\mathbf{A} = \mathbf{A}\boldsymbol{\Lambda}.$$

It is well known that $\hat{\mathbf{W}}\hat{\boldsymbol{\Theta}}'\mathbf{E}'\mathbf{H}_n\mathbf{X}$ and $\hat{\boldsymbol{\Theta}}'\mathbf{E}'\mathbf{H}_n\mathbf{X}\hat{\mathbf{W}}$ have the same nonzero eigenvalues. Moreover, $\hat{\boldsymbol{\Theta}}'\mathbf{E}'\mathbf{H}_n\mathbf{X}\mathbf{A}$ is the eigenvector matrix of $\hat{\boldsymbol{\Theta}}'\mathbf{E}'\mathbf{H}_n\mathbf{X}\hat{\mathbf{W}}$. We thus establish the relationship between $\mathbf{A}$ in the penalized LDA and $\mathbf{W}$ in the penalized optimal scoring model (1).

## 3 Optimal Scoring for Unsupervised Learning

In this section we extend the notion of optimal scoring to unsupervised learning problems, leading to a new framework for dimensionality reduction and clustering analysis simultaneously.

### 3.1 Framework

In particular, we relax $\mathbf{E}\boldsymbol{\Theta}$ in (1) as a sample scoring matrix $\mathbf{Y}$ and define the following penalized model:

$$\min_{\mathbf{Y},\mathbf{W}} \left\{ f(\mathbf{Y},\mathbf{W}) \triangleq \frac{1}{2}\|\mathbf{Y} - \mathbf{H}_n\mathbf{X}\mathbf{W}\|_F^2 + \frac{\sigma^2}{2}\mathrm{tr}(\mathbf{W}'\mathbf{W}) \right\} \tag{2}$$

under the constraints $\mathbf{1}_n'\mathbf{Y} = \mathbf{0}$ and $\mathbf{Y}'\mathbf{Y} = \mathbf{I}_{c-1}$. The following theorem provides a solution for this problem.

**Theorem 2** *A minimizer of Problem (2) is* $\hat{\mathbf{Y}}$ *and* $\hat{\mathbf{W}} = (\mathbf{X}'\mathbf{H}_n\mathbf{X} + \sigma^2\mathbf{I}_p)^{-1}\mathbf{X}'\mathbf{H}_n\hat{\mathbf{Y}}$, *where* $\hat{\mathbf{Y}}$ *is the* $n \times (c-1)$ *orthogonal matrix of the top eigenvectors of* $\mathbf{H}_n\mathbf{X}(\mathbf{X}'\mathbf{H}_n\mathbf{X} + \sigma^2\mathbf{I}_p)^{-1}\mathbf{X}'\mathbf{H}_n$.

The proof is given in Appendix A. Note that all the eigenvalues of $\mathbf{H}_n\mathbf{X}(\mathbf{X}'\mathbf{H}_n\mathbf{X} + \sigma^2\mathbf{I}_p)^{-1}\mathbf{X}'\mathbf{H}_n$ are between 0 and 1. Especially, when $\sigma^2 = \mathbf{0}$, the eigenvalues are either 1 or 0. In this case, if $\mathrm{rk}(\mathbf{H}_n\mathbf{X}) \geq c-1$, $f(\hat{\mathbf{Y}}, \hat{\mathbf{W}})$ achieves its minimum 0, otherwise the minimum value is $\frac{c-1-\mathrm{rk}(\mathbf{H}_n\mathbf{X})}{2}$.

With the estimates of $\mathbf{Y}$ and $\mathbf{W}$, we can develop an unsupervised learning procedure. It is clear that $\mathbf{W}$ can be treated as a non-orthogonal projection matrix and $\mathbf{H}_n\mathbf{X}\mathbf{W}$ is then the low-dimensional configuration of $\mathbf{X}$. Using this treatment, we obtain a new alternative to the regression formulation of PCA by Zou *et al.* [17]. In this paper, however, we concentrate on the application of the framework in clustering analysis.

### 3.2 Optimal Discriminant Clustering

Our clustering procedure is given in Algorithm 1. We refer to this procedure as *optimal discriminant clustering* due to its relationship with LDA, which is shown by the connection between (1) and (2).

Assume that $\tilde{\mathbf{X}} = [\tilde{\mathbf{x}}_1, \ldots, \tilde{\mathbf{x}}_n]'$ ($n \times r$) is a feature matrix corresponding to the data matrix $\mathbf{X}$. In this case, we have

$$\mathbf{S} = \mathbf{H}_n\tilde{\mathbf{X}}(\tilde{\mathbf{X}}'\mathbf{H}_n\tilde{\mathbf{X}} + \sigma^2\mathbf{I}_r)^{-1}\tilde{\mathbf{X}}'\mathbf{H}_n = \mathbf{C}(\mathbf{C} + \sigma^2\mathbf{I}_n)^{-1},$$

where $\mathbf{C} = \mathbf{H}_n\tilde{\mathbf{X}}\tilde{\mathbf{X}}'\mathbf{H}_n$ is the $n \times n$ centered kernel matrix. This implies that we can obtain $\hat{\mathbf{Y}}$ without the explicit use of the feature matrix $\tilde{\mathbf{X}}$. Moreover, we can compute $\mathbf{Z}$ by

$$\mathbf{Z} = \mathbf{H}_n\tilde{\mathbf{X}}(\tilde{\mathbf{X}}'\mathbf{H}_n\tilde{\mathbf{X}} + \sigma^2\mathbf{I}_r)^{-1}\tilde{\mathbf{X}}'\mathbf{H}_n\mathbf{Y} = \mathbf{S}\mathbf{Y}.$$

We are thus able to devise this clustering algorithm by using the reproducing kernel $k(\cdot, \cdot)$ : $\mathcal{X} \times \mathcal{X} \to \mathbb{R}$ such that $K(\mathbf{x}_i, \mathbf{x}_j) = \tilde{\mathbf{x}}_i'\tilde{\mathbf{x}}_j$ and $\mathbf{K} = \tilde{\mathbf{X}}\tilde{\mathbf{X}}'$.

---

**Algorithm 1** Optimal Discriminant Clustering Algorithm

---

1: **procedure** ODC($\mathbf{H}_n\mathbf{X}, c, \sigma^2$)
2:     Estimate $\hat{\mathbf{Y}}$ and $\hat{\mathbf{W}}$ according to Theorem 2;
3:     Calculate $\mathbf{Z} = [\mathbf{z}_1, \ldots, \mathbf{z}_n]' = \mathbf{H}_n\mathbf{X}\hat{\mathbf{W}}$;
4:     Perform $K$-means on the $\mathbf{z}_i$;
5:     Return the partition of the $\mathbf{z}_i$ as the partition of the $\mathbf{x}_i$.
6: **end procedure**

---

### 3.3 Related Work

We now explore the connection of the optimal discriminant clustering with the discriminative clustering algorithm [3] and spectral clustering [7]. Recall that $\hat{\mathbf{Y}}$ is the matrix of the $c-1$ top eigenvectors of $\mathbf{C}(\mathbf{C} + \sigma^2\mathbf{I}_n)^{-1}$. Consider that if $\lambda \neq 0$ is an eigenvalue of $\mathbf{C}$ with associated eigenvector $\mathbf{u}$, then $\lambda/(\lambda + \sigma^2) (\neq 0)$ is an eigenvalue of $\mathbf{C}(\mathbf{C} + \sigma^2\mathbf{I}_n)^{-1}$ with associated eigenvector $\mathbf{u}$. Moreover, $\lambda/(\lambda + \sigma^2)$ is increasing as $\lambda$ increases. This implies that $\hat{\mathbf{Y}}$ is also the matrix of the $c-1$ top eigenvectors of $\mathbf{C}$. As we know, the spectral clustering applies a rounding scheme such as $K$-means directly on $\hat{\mathbf{Y}}$. We thus have a relationship between the spectral clustering and optimal discriminant clustering.

We study the relationship between the discriminative clustering algorithm and the spectral clustering algorithm. Let $\mathbf{M}$ be a linear transformation from the $r$-dimensional $\tilde{\mathbf{X}}$ to an $s$-dimensional transformed feature space $\mathbf{F}$, namely

$$\mathbf{F} = \tilde{\mathbf{X}}\mathbf{M},$$

where $\mathbf{M}$ is an $r \times s$ matrix of rank $s$ ($s < r$). The corresponding scatter matrices in the $\mathbf{F}$-space are thus given by $\mathbf{M}'\boldsymbol{\Sigma}\mathbf{M}$ and $\mathbf{M}'\boldsymbol{\Sigma}_b\mathbf{M}$. The discriminative clustering algorithm [3, 13] in the reproducing kernel Hilbert space (RKHS) tries to solve the problem of

$$\underset{\mathbf{E}, \mathbf{M}}{\operatorname{argmax}} \; f(\mathbf{E}, \mathbf{M}) \triangleq \operatorname{tr}((\mathbf{M}'(\boldsymbol{\Sigma} + \sigma^2\mathbf{I}_r)\mathbf{M})^{-1}\mathbf{M}'\boldsymbol{\Sigma}_b\mathbf{M})$$

$$= \operatorname{tr}\big((\mathbf{M}'(\tilde{\mathbf{X}}'\mathbf{H}_n\tilde{\mathbf{X}} + \sigma^2\mathbf{I}_r)\mathbf{M})^{-1}\mathbf{M}'\tilde{\mathbf{X}}'\mathbf{H}_n\mathbf{E}(\mathbf{E}'\mathbf{E})^{-1}\mathbf{E}'\mathbf{H}_n\tilde{\mathbf{X}}\mathbf{M}\big)$$

Applying the discussion in [15] to $\mathbf{H}_n\tilde{\mathbf{X}}\mathbf{M}(\mathbf{M}'(\tilde{\mathbf{X}}'\mathbf{H}_n\tilde{\mathbf{X}} + \sigma^2\mathbf{I}_r)\mathbf{M})^{-1}\mathbf{M}'\tilde{\mathbf{X}}'\mathbf{H}_n$, we have the following relaxation problem

$$\max_{\substack{\mathbf{Y} \in \mathbb{R}^{n \times (c-1)}, \mathbf{M} \in \mathbb{R}^{r \times s}}} \operatorname{tr}(\mathbf{Y}'\mathbf{H}_n\tilde{\mathbf{X}}\mathbf{M}(\mathbf{M}'(\tilde{\mathbf{X}}'\mathbf{H}_n\tilde{\mathbf{X}} + \sigma^2\mathbf{I}_r)\mathbf{M})^{-1}\mathbf{M}'\tilde{\mathbf{X}}'\mathbf{H}_n\mathbf{Y}), \quad (3)$$
$$\text{s.t. } \mathbf{Y}'\mathbf{Y} = \mathbf{I}_{c-1} \text{ and } \mathbf{Y}'\mathbf{1}_n = \mathbf{0}.$$

Express $\mathbf{M} = \tilde{\mathbf{X}}'\mathbf{H}_n\mathbf{B} + \mathbf{N}$ where $\mathbf{N}$ satisfies $\mathbf{N}'\tilde{\mathbf{X}}'\mathbf{H}_n = \mathbf{0}$ (i.e., $\mathbf{N} \in span\{\tilde{\mathbf{X}}'\mathbf{H}_n\}^\perp$) and $\mathbf{B}$ is some $n \times s$ matrix. Under the condition of either $\sigma^2 = 0$ or $\mathbf{N} = \mathbf{0}$ (i.e., $\mathbf{M} \in span\{\tilde{\mathbf{X}}'\mathbf{H}_n\}$), we can obtain that

$$\mathbf{H}_n\tilde{\mathbf{X}}\mathbf{M}(\mathbf{M}'(\tilde{\mathbf{X}}'\mathbf{H}_n\tilde{\mathbf{X}} + \sigma^2\mathbf{I}_r)\mathbf{M})^{-1}\mathbf{M}'\tilde{\mathbf{X}}'\mathbf{H}_n = \mathbf{C}\mathbf{B}(\mathbf{B}'(\mathbf{C}\mathbf{C} + \sigma^2\mathbf{C})\mathbf{B})^{-1}\mathbf{B}'\mathbf{C}.$$

Again consider that if $\lambda \neq 0$ is an eigenvalue of $\mathbf{C}$ with associated eigenvector $\mathbf{u}$, then $\lambda/(\lambda + \sigma^2) \neq 0$ is an eigenvalue of $\mathbf{C}(\mathbf{C}\mathbf{C} + \sigma^2\mathbf{C})^+\mathbf{C}$ with associated eigenvector $\mathbf{u}$. Moreover, $\lambda/(\lambda + \sigma^2)$ is increasing in $\lambda$. We now directly obtain the following theorem from Theorem 3.1 in [13].

**Theorem 3** *Let $\mathbf{Y}^*$ and $\mathbf{M}^*$ be the solution of Problem (3). Then*

Table 1: Summary of the benchmark datasets, where $c$ is the number of classes, $p$ is the dimension of the input vector, and $n$ is the number of samples in the dataset.

| Types | Dataset | $c$ | $p$ | $n$ |
|---|---|---|---|---|
| Face | ORL | 40 | 1024 | 400 |
| | Yale | 15 | 1024 | 165 |
| | PIE | 68 | 1024 | 6800 |
| Gene | SRBCT | 4 | 2308 | 63 |
| UCI | Iris | 4 | 4 | 150 |
| | Yeast | 10 | 8 | 1484 |
| | Image segmentation | 7 | 19 | 2100 |
| | Statlog landsat satellite | 7 | 36 | 2000 |

(i) *If $\sigma^2 = 0$, $\mathbf{Y}^*$ is the solution of the following problem*

$$\operatorname{argmax}_{\mathbf{Y} \in \mathbb{R}^{n \times (c-1)}} \ \operatorname{tr}(\mathbf{Y}'\mathbf{C}\mathbf{C}^+\mathbf{Y}),$$
$$s.t. \ \mathbf{Y}'\mathbf{Y} = \mathbf{I}_{c-1} \ and \ \mathbf{Y}'\mathbf{1}_n = \mathbf{0}.$$

(ii) *If $\mathbf{M} \in span\{\tilde{\mathbf{X}}'\mathbf{H}_n\}$, $\mathbf{Y}^*$ is the solution of the following problem:*

$$\operatorname{argmax}_{Y \in \mathbb{R}^{n \times (c-1)}} \ \operatorname{tr}(\mathbf{Y}'\mathbf{C}\mathbf{Y}),$$
$$s.t. \ \mathbf{Y}'\mathbf{Y} = \mathbf{I}_{c-1} \ and \ \mathbf{Y}'\mathbf{1}_n = \mathbf{0}.$$

Theorem 3 shows that discriminative clustering is essentially equivalent to spectral clustering. This further leads us to a relationship between the discriminative clustering and optimal discriminant clustering from the relationship between the spectral clustering and optimal discriminant clustering. In summary, we are able to unify the discriminative clustering as well as spectral clustering into the optimal scoring framework in (2).

## 4 Experimental Study

To evaluate the performance of our optimal discriminant clustering (ODC) algorithm, we conducted experimental comparisons with other related clustering algorithms on several real-world datasets. In particular, the comparison was implemented on three face datasets, the "SRBCT" gene dataset, and four UCI datasets. Further details of these datasets are summarized in Table 1.

To effectively evaluate the performance, we employed two typical measurements: the *Normalized Mutual Information* (NMI) and the *Clustering Error* (CE). It should be mentioned that for NMI, the larger this value, the better the performance. For CE, the smaller the value, the better the performance. More details and the corresponding implementations for both can be found in [11].

In the experiments, we compared our ODC with four different clustering algorithms, i.e., the conventional $K$-means [1], normalized cut (NC) [9], DisCluster [3] and DisKmeans [13]. It is worth noting that two discriminative clustering algorithms: DisCluster [3] and DisKmeans [13], are very closely related to our ODC, because they are derived from the discriminant analysis criteria in essence (also see the analysis in Section 3.3). In addition, the implementation code for NC is available at http://www.cis.upenn.edu/~jshi/software/. For the sake of simplicity, the parameter $\sigma^2$ in ODC is sought from the range $\sigma^2 \in \{10^{-3}, 10^{-2.5}, 10^{-2}, 10^{-1.5}, 10^{-1}, 10^{-0.5}, 10^0, 10^{0.5}, 10^1, 10^{1.5}, 10^2, 10^{2.5}, 10^3\}$. Similarly, the parameters in other clustering algorithms compared here are also searched in a wide range.

For simplicity, we just reported the best results of clustering algorithms with respect to different parameters on each dataset. Table 2 summaries the NMI and CE on all datasets. According to the NMI values in Table 2, our ODC outperforms other clustering algorithms on five datasets: ORL, SRBCT, iris, yeast and image segmentation. According to the CE values in Table 2, it is obvious that the performance of our ODC is best in comparison with other algorithms on all the datasets, and NC and DisKmeans algorithms can achieve the almost same performance with ODC on the SRBCT and iris datasets respectively. Also, it is seen that the DisCluster algorithm has dramatically different performance based on the NMI and CE. The main reason is that the final solution in DisCluster is very sensitive to the initial variables and numerical computation.

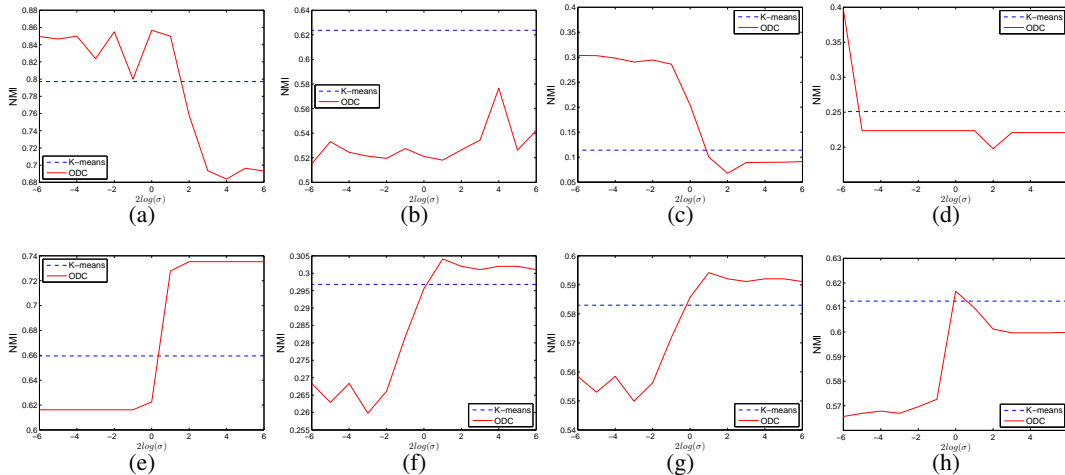

Figure 1: The NMI versus the parameter $\sigma$ tuning in ODC on all datasets, where the NMI of $K$-means is used as the baseline: (a) `ORL`; (b) `Yale`; (c) `PIE`; (d) `SRBCT`; (e) `iris`; (f) `yeast`; (g) `image segmentation`; (h) `statlog landsat satellite`.

In order to reveal the effect of the parameter $\sigma$ on ODC, Figures 1 and 2 depict the NMI and CE results of ODC with respect to different parameters $\sigma$ on all datasets. Similar to [11, 13], we used the results of $K$-means as a baseline. From Figures 1 and 2, we can see that similar to the conventional clustering algorithms (including the compared algorithms), the parameter $\sigma$ has a significant impact on the performance of ODC, especially when the evaluation results are measured by NMI. In contrast to the result in Figure 1, the effect of the parameter $\sigma$ becomes less pronounced in Figure 2.

Table 2: Clustering results: the *Normalized Mutual Information* (NMI) and the *Clustering Error* (CE) (%) of all clustering algorithms are calculated on different datasets.

| Measure | Dataset | $K$-means | NC | DisCluster | DisKmeans | ODC |
|---|---|---|---|---|---|---|
| NMI | ORL | 0.7971 | 0.8015 | 0.7978 | 0.8531 | **0.8567** |
| | Yale | **0.6237** | 0.6203 | 0.5974 | 0.5641 | 0.5766 |
| | PIE | 0.1140 | 0.2232 | 0.1940 | **0.3360** | 0.3035 |
| | SRBCT | 0.2509 | 0.3722 | 0.3216 | 0.2683 | **0.3966** |
| | Iris | 0.6595 | 0.6876 | 0.7248 | **0.7353** | **0.7353** |
| | Yeast | 0.2968 | 0.2915 | 0.2993 | 0.3020 | **0.3041** |
| | Image segmentation | 0.5830 | 0.5500 | 0.5700 | 0.5934 | **0.5942** |
| | Statlog landsat satellite | 0.6126 | **0.6316** | 0.6152 | 0.6009 | 0.6166 |
| CE (%) | ORL | 38.25 | 34.50 | 38.75 | 29.00 | **28.50** |
| | Yale | 45.45 | 46.06 | 45.45 | 45.45 | **44.84** |
| | PIE | 79.82 | 79.82 | 77.35 | 66.23 | **65.52** |
| | SRBCT | 55.55 | **47.61** | 50.79 | 53.96 | **47.61** |
| | Iris | 16.66 | 15.33 | 12.66 | **11.33** | **11.33** |
| | Yeast | 59.43 | 59.90 | 59.43 | 57.07 | **56.73** |
| | Image segmentation | 45.14 | 49.47 | 45.95 | 41.66 | **40.23** |
| | Statlog landsat satellite | 32.30 | 32.65 | 32.25 | 31.20 | **30.50** |

## 5    Concluding Remarks

In this paper we have proposed a regression framework to deal with unsupervised dimensionality reduction and clustering simultaneously. The framework is based on the optimal scoring and ridge penalty. In particular, we have developed a new clustering algorithm which is called *optimal discriminant clustering* (ODC). ODC can efficiently identify the optimal solution and it has an underlying relationship with the discriminative clustering and spectral clustering.

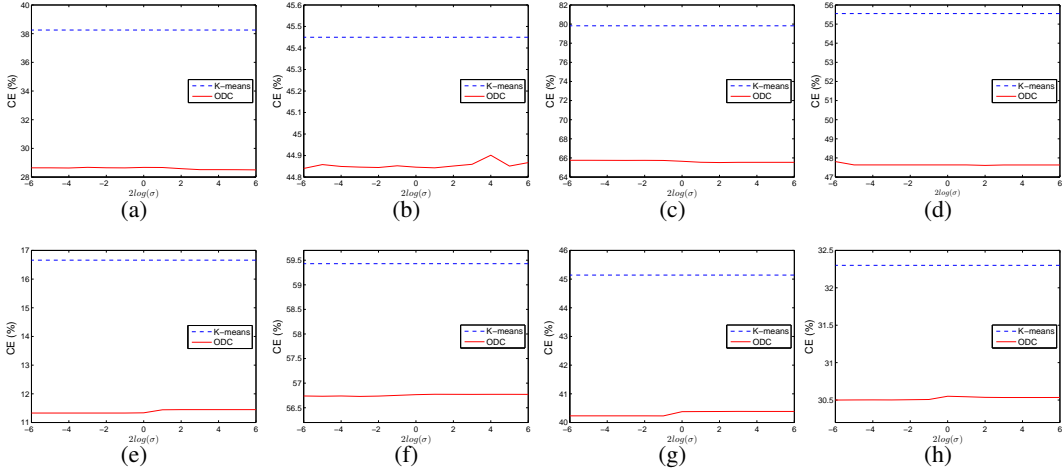

Figure 2: The CE (%) versus the parameter $\sigma$ tuning in ODC on all datasets, where the CE (%) of $K$-means is used as the baseline: (a) ORL; (b) Yale; (c) PIE; (d) SRBCT; (e) iris; (f) yeast; (g) image segmentation; (h) statlog landsat satellite.

This framework allows us for developing a sparse unsupervised learning algorithm; that is, we alternatively consider the following optimization problem:

$$\min_{\mathbf{Y},\,\mathbf{W}} f(\mathbf{Y}, \mathbf{W}) = \frac{1}{2}\|\mathbf{Y} - \mathbf{H}_n\mathbf{X}\mathbf{W}\|_F^2 + \frac{\lambda_1}{2}\mathrm{tr}(\mathbf{W}'\mathbf{W}) + \lambda_2\|\mathbf{W}\|_1$$

under the constraints $\mathbf{1}_n'\mathbf{Y} = \mathbf{0}$ and $\mathbf{Y}'\mathbf{Y} = \mathbf{I}_{c-1}$. We will study this further.

**Acknowledgement**

This work has been supported in part by program for Changjiang Scholars and Innovative Research Team in University (IRT0652, PCSIRT), China.

## A Proof of Theorem 2

For simplicity, we replace $\mathbf{H}_n\mathbf{X}$ by $\mathbf{X}$ and let $q = c-1$ in the following derivation. Consider the Lagrange function:

$$L(\mathbf{Y}, \mathbf{W}, \mathbf{B}, \mathbf{b})$$
$$= \frac{1}{2}\mathrm{tr}(\mathbf{Y}'\mathbf{Y}) - \mathrm{tr}(\mathbf{Y}'\mathbf{X}\mathbf{W}) + \frac{1}{2}\mathrm{tr}(\mathbf{W}'(\mathbf{X}'\mathbf{X}+\sigma^2\mathbf{I}_p)\mathbf{W}) - \frac{1}{2}\mathrm{tr}(\mathbf{B}(\mathbf{Y}'\mathbf{Y}-\mathbf{I}_q)) - \mathrm{tr}(\mathbf{b}'\mathbf{Y}'\mathbf{1}_n),$$

where $\mathbf{B}$ is a $q{\times}q$ symmetric matrix of Lagrange multipliers and $\mathbf{b}$ is a $q{\times}1$ vector of Lagrange multipliers. By direct differentiation, it can be shown that

$$\frac{\partial L}{\partial \mathbf{Y}} = \mathbf{Y} - \mathbf{X}\mathbf{W} - \mathbf{Y}\mathbf{B} - \mathbf{1}_n\mathbf{b}',$$

$$\frac{\partial L}{\partial \mathbf{W}} = (\mathbf{X}'\mathbf{X} + \sigma^2\mathbf{I}_p)\mathbf{W} - \mathbf{X}'\mathbf{Y}.$$

Letting $\frac{\partial L}{\partial \mathbf{Y}} = \mathbf{0}$, we have

$$\mathbf{Y} - \mathbf{X}\mathbf{W} - \mathbf{Y}\mathbf{B} - \mathbf{1}_n\mathbf{b}' = \mathbf{0}.$$

Pre-multiplying both sides of the above equation by $\mathbf{1}_n'$, we obtain $\mathbf{b} = 0$. Thus, it follows from $\frac{\partial L}{\partial \mathbf{Y}} = \mathbf{0}$ and $\frac{\partial L}{\partial \mathbf{W}} = \mathbf{0}$ that

$$\begin{cases} \mathbf{Y} - \mathbf{X}\mathbf{W} - \mathbf{Y}\mathbf{B} = \mathbf{0}, \\ \mathbf{W} = (\mathbf{X}'\mathbf{X} + \sigma^2\mathbf{I}_p)^{-1}\mathbf{X}'\mathbf{Y}. \end{cases}$$

Substituting the second equation into the first equation, we further have
$$(\mathbf{I}_n - \mathbf{X}(\mathbf{X}'\mathbf{X} + \sigma^2\mathbf{I}_p)^{-1}\mathbf{X}')\mathbf{Y} = \mathbf{Y}\mathbf{B}.$$

Now we take the spectral decomposition of $\mathbf{B}$ as $\mathbf{B} = \mathbf{U}_B\mathbf{\Lambda}_B\mathbf{U}_B'$ where $\mathbf{U}_B$ is a $q\times q$ orthonormal matrix and $\mathbf{\Lambda}_B$ is a $q\times q$ diagonal matrix. We thus have $(\mathbf{I}_n - \mathbf{X}(\mathbf{X}'\mathbf{X} + \sigma^2\mathbf{I}_p)^{-1}\mathbf{X}')\mathbf{Y}\mathbf{U}_B = \mathbf{Y}\mathbf{U}_B\mathbf{\Lambda}_B$. This shows that the diagonal entries of $\mathbf{\Lambda}_B$ and the columns of $\mathbf{Y}\mathbf{U}_B$ are the eigenvalues and the associated eigenvectors of $\mathbf{I}_n - \mathbf{X}(\mathbf{X}'\mathbf{X} + \sigma^2\mathbf{I}_p)^{-1}\mathbf{X}'$.

We consider the case that $n \geq p$. Let the SVD of $\mathbf{X}$ be $\mathbf{X} = \mathbf{U}\mathbf{\Gamma}\mathbf{V}'$ where $\mathbf{U}$ $(n\times p)$ and $\mathbf{V}$ $(p\times p)$ are orthogonal, and $\mathbf{\Gamma} = \mathrm{diag}(\gamma_1,\ldots,\gamma_p)$ $(p\times p)$ is a diagonal matrix with $\gamma_1 \geq \gamma_2 \geq \cdots \geq \gamma_p \geq 0$. We then have $\mathbf{X}(\mathbf{X}'\mathbf{X} + \sigma^2\mathbf{I}_p)^{-1}\mathbf{X}' = \mathbf{U}\mathbf{\Lambda}\mathbf{U}'$, where $\mathbf{\Lambda} = \mathrm{diag}(\lambda_1,\ldots,\lambda_p)$ with $\lambda_i = \gamma_i^2/(\gamma_i^2 + \sigma^2)$. There exists such an $n\times(n-p)$ orthogonal matrix $\mathbf{U}_3$ that its last column is $\frac{1}{\sqrt{n}}\mathbf{1}_n$ and $[\mathbf{U}, \mathbf{U}_3]$ is an $n\times n$ orthonormal matrix. That is, $\mathbf{U}_3$ is the eigenvector matrix of $\mathbf{X}(\mathbf{X}'\mathbf{X} + \sigma^2\mathbf{I}_p)^{-1}\mathbf{X}'$ corresponding to the eigenvalue 0. Let $\mathbf{U}_1$ be the $n\times q$ matrix of the first $q$ columns of $[\mathbf{U}, \mathbf{U}_3]$.

We now define $\hat{\mathbf{Y}} = \mathbf{U}_1$, $\hat{\mathbf{W}} = (\mathbf{X}'\mathbf{X} + \sigma^2\mathbf{I}_p)^{-1}\mathbf{X}'\mathbf{U}_1$, $\mathbf{U}_B = \mathbf{I}_q$ and $\mathbf{\Lambda}_B = \mathrm{diag}(1 - \lambda_1, \ldots, 1 - \lambda_q)$ where $\lambda_i = 0$ whenever $i > p$. It is easily seen that such a $\hat{\mathbf{Y}}$ satisfies $\hat{\mathbf{Y}}'\hat{\mathbf{Y}} = \mathbf{I}_q$ and $\hat{\mathbf{Y}}'\mathbf{1}_n = \mathbf{0}$ due to $\mathbf{U}_1'\mathbf{U}_1 = \mathbf{I}_q$ and $\mathbf{X}'\mathbf{1}_n = \mathbf{0}$. Moreover, we have
$$f(\hat{\mathbf{Y}}, \hat{\mathbf{W}}) = \frac{q}{2} - \frac{1}{2}\sum_{i=1}^q \lambda_i = \frac{q}{2} - \frac{1}{2}\sum_{i=1}^q \frac{\gamma_i^2}{\gamma_i^2 + \sigma^2}$$
where $\gamma_i = 0$ whenever $i > p$. Note that all the eigenvalues of $\mathbf{X}(\mathbf{X}'\mathbf{X} + \sigma^2\mathbf{I}_p)^{-1}\mathbf{X}'$ are between 0 and 1. Especially, when $\sigma^2 = \mathbf{0}$, the eigenvalues are either 1 or 0. In this case, if $\mathrm{rk}(\mathbf{X}) \geq q$, $f(\hat{\mathbf{Y}}, \hat{\mathbf{W}})$ achieves its minimum 0, otherwise the minimum value is $\frac{q-\mathrm{rk}(\mathbf{X})}{2}$.

To verify that $(\hat{\mathbf{Y}}, \hat{\mathbf{W}})$ is a minimizer of problem (2), we consider the Hessian matrix of $L$ with respect to $(\mathbf{Y}, \mathbf{W})$. Let $\mathrm{vec}(\mathbf{Y}') = (y_{11},\ldots,y_{1q},y_{21},\ldots,y_{nq})'$ and $\mathrm{vec}(\mathbf{W}') = (w_{11},\ldots,w_{1q},w_{21},\ldots,w_{pq})'$. The Hessian matrix is then given by
$$\mathcal{H}(\mathbf{Y},\mathbf{W}) = \begin{bmatrix} \frac{\partial^2 L}{\partial\mathrm{vec}(\mathbf{Y}')\partial\mathrm{vec}(\mathbf{Y}')'} & \frac{\partial^2 L}{\partial\mathrm{vec}(\mathbf{Y}')\partial\mathrm{vec}(\mathbf{W}')'} \\ \frac{\partial^2 L}{\partial\mathrm{vec}(\mathbf{W}')\partial\mathrm{vec}(\mathbf{Y}')'} & \frac{\partial^2 L}{\partial\mathrm{vec}(\mathbf{W}')\partial\mathrm{vec}(\mathbf{W}')'} \end{bmatrix} = \begin{bmatrix} (\mathbf{I}_q-\mathbf{B})\otimes\mathbf{I}_n & -\mathbf{I}_q\otimes\mathbf{X} \\ -\mathbf{I}_q\otimes\mathbf{X}' & \mathbf{I}_q\otimes(\mathbf{X}'\mathbf{X}+\sigma^2\mathbf{I}_p) \end{bmatrix}.$$

Let $\mathbf{C}' = [\mathbf{C}_1', \mathbf{C}_2']$, where $\mathbf{C}_1$ and $\mathbf{C}_2$ are $n\times q$ and $p\times q$, be an arbitrary nonzero $(n+p)\times q$ matrix such that $\mathbf{C}_1'[\mathbf{1}_n, \hat{\mathbf{Y}}] = \mathbf{0}$, which is equivalent to $\mathbf{C}_1'\mathbf{1}_n = \mathbf{0}$ and $\mathbf{C}_1'\mathbf{U}_1 = \mathbf{0}$.

If $\mathrm{rk}(\mathbf{X}) \leq q$, we have $\mathbf{C}_1'\mathbf{X} = \mathbf{0}$. Hence,
$$\mathrm{vec}(\mathbf{C}')'\mathcal{H}(\hat{\mathbf{Y}}, \hat{\mathbf{W}})\mathrm{vec}(\mathbf{C}') = \mathrm{tr}(\mathbf{C}_1'\mathbf{C}_1(\mathbf{I}_q - \mathbf{B})) - 2\mathrm{tr}(\mathbf{C}_1'\mathbf{X}\mathbf{C}_2) + \mathrm{tr}(\mathbf{C}_2'(\mathbf{X}'\mathbf{X} + \sigma^2\mathbf{I}_p)\mathbf{C}_2)$$
$$= \mathrm{tr}(\mathbf{C}_1'\mathbf{C}_1(\mathbf{I}_q - \mathbf{B})) + \mathrm{tr}(\mathbf{C}_2'(\mathbf{X}'\mathbf{X} + \sigma^2\mathbf{I}_p)\mathbf{C}_2) \geq 0.$$

This implies that $(\hat{\mathbf{Y}}, \hat{\mathbf{W}})$ is a minimizer of problem (2).

In the case that $\mathrm{rk}(\mathbf{X}) = m > q$, we have $p > q$. Thus we can partition $\mathbf{U}$ and $\mathbf{V}$ into $\mathbf{U} = [\mathbf{U}_1, \mathbf{U}_2]$ and $\mathbf{V} = [\mathbf{V}_1, \mathbf{V}_2]$ where $\mathbf{V}_1$ and $\mathbf{V}_2$ are $p\times q$ and $p\times(p-q)$. Thus,
$$\begin{aligned}\mathrm{vec}(\mathbf{C}')'\mathcal{H}(\hat{\mathbf{Y}}, \hat{\mathbf{W}})\mathrm{vec}(\mathbf{C}') &= \mathrm{tr}(\mathbf{C}_1'\mathbf{C}_1(\mathbf{I}_q - \mathbf{B})) - 2\mathrm{tr}(\mathbf{C}_1'\mathbf{X}\mathbf{C}_2) + \mathrm{tr}(\mathbf{C}_2'(\mathbf{X}'\mathbf{X} + \sigma^2\mathbf{I}_p)\mathbf{C}_2) \\ &\geq \mathrm{tr}(\mathbf{C}_1'\mathbf{U}_2\mathbf{\Lambda}_2\mathbf{U}_2'\mathbf{C}_1) - 2\mathrm{tr}(\mathbf{C}_1'\mathbf{U}_2\mathbf{\Gamma}_2\mathbf{V}_2'\mathbf{C}_2) + \mathrm{tr}(\mathbf{C}_2'\mathbf{V}_2\mathbf{D}_2\mathbf{V}_2'\mathbf{C}_2) \\ &\quad + \mathrm{tr}(\mathbf{C}_1'\mathbf{U}_3\mathbf{U}_3'\mathbf{C}_1\mathbf{\Lambda}_1) + \mathrm{tr}(\mathbf{C}_2'\mathbf{V}_1\mathbf{D}_1\mathbf{V}_1'\mathbf{C}_2) \\ &= \mathrm{tr}\big[(\mathbf{\Lambda}_2^{1/2}\mathbf{U}_2'\mathbf{C}_1 - \mathbf{D}_2^{1/2}\mathbf{V}_2'\mathbf{C}_2)'(\mathbf{\Lambda}_2^{1/2}\mathbf{U}_2'\mathbf{C}_1 - \mathbf{D}_2^{1/2}\mathbf{V}_2'\mathbf{C}_2)\big] \\ &\quad + \mathrm{tr}(\mathbf{C}_1'\mathbf{U}_3\mathbf{U}_3'\mathbf{C}_1\mathbf{\Lambda}_1) + \mathrm{tr}(\mathbf{C}_2'\mathbf{V}_1\mathbf{D}_1\mathbf{V}_1'\mathbf{C}_2) \geq 0.\end{aligned}$$

Here $\mathbf{\Lambda}_1 = \mathrm{diag}(\lambda_1, \ldots, \lambda_q)$, $\mathbf{\Lambda}_2 = \mathrm{diag}(\lambda_{q+1}, \ldots, \lambda_p)$, $\mathbf{\Gamma}_1 = \mathrm{diag}(\gamma_1, \ldots, \gamma_q)$, $\mathbf{\Gamma}_2 = \mathrm{diag}(\gamma_{q+1}, \ldots, \gamma_p)$, $\mathbf{D}_1 = \mathbf{\Gamma}_1^2 + \sigma^2\mathbf{I}_q$ and $\mathbf{D}_2 = \mathbf{\Gamma}_2^2 + \sigma^2\mathbf{I}_{p-q}$, so we have $\mathbf{\Gamma}_2 = \mathbf{D}_2^{1/2}\mathbf{\Lambda}_2^{1/2}$. Moreover, we use the fact that
$$\mathrm{tr}(\mathbf{C}_1'\mathbf{U}_2\mathbf{U}_2'\mathbf{C}_1\mathbf{\Lambda}_1) \geq \mathrm{tr}(\mathbf{C}_1'\mathbf{U}_2\mathbf{\Lambda}_2\mathbf{U}_2'\mathbf{C}_1)$$
because $\lambda_i\mathbf{I}_q - \mathbf{\Lambda}_2$ for $i = 1, \ldots, q$ are positive semidefinite.

If $n < p$, we also make the SVD of $\mathbf{X}$ as $\mathbf{X} = \mathbf{U}\mathbf{\Gamma}\mathbf{V}'$. But, right now, $\mathbf{U}$ is $n\times n$, $\mathbf{V}$ is $n\times p$, and $\mathbf{\Lambda}$ is $n\times n$. Using this SVD, we have the same result as the case of $n \geq p$.

# References

[1] C. M. Bishop. *Pattern Recognition and Machine Learning*. Springer, first edition, 2007.

[2] L. Clemmensen, T. Hastie, and B. Erbøll. Sparse discriminant analysis. Technical report, June 2008.

[3] F. De la Torre and T. Kanade. Discriminative cluster analysis. In *The 23rd International Conference on Machine Learning*, 2006.

[4] R. O. Duda, P. E. Hart, and D. G. Stork. *Pattern Classification*. John Wiley and Sons, New York, second edition, 2001.

[5] T. Hastie, A. Buja, and R. Tibshirani. Penalized discriminant analysis. *The Annals of Statistics*, 23(1):73–102, 1995.

[6] T. Hastie, R. Tibshirani, and A. Buja. Flexible discriminant analysis by optimal scoring. *Journal of the American Statistical Association*, 89(428):1255–1270, 1994.

[7] A. Y. Ng, M. I. Jordan, and Y. Weiss. On spectral clustering: analysis and an algorithm. In *Advances in Neural Information Processing Systems 14*, volume 14, 2002.

[8] C. H. Park and H. Park. A relationship between linear discriminant analysis and the generalized minimum squared error solution. *SIAM Journal on Matrix Analysis and Applications*, 27(2):474–492, 2005.

[9] J. Shi and J. Malik. Normalized cuts and image segmentation. *IEEE Transactions on Pattern Analysis and Machine Intelligence*, 22(8):888–905, 2000.

[10] R. Tibshirani. Regression shrinkage and selection via the lasso. *Journal of the Royal Statistical Society, Series B*, 58:267–288, 1996.

[11] M. Wu and B. Schölkopf. A local learning approach for clustering. In *Advances in Neural Information Processing Systems 19*, 2007.

[12] J. Ye. Least squares linear discriminant analysis. In *The Twenty-Fourth International Conference on Machine Learning*, 2007.

[13] J. Ye, Z. Zhao, and M. Wu. Discriminative k-means for clustering. In *Advances in Neural Information Processing Systems 20*, 2008.

[14] Z. Zhang, G. Dai, and M. I. Jordan. A flexible and efficient algorithm for regularized Fisher discriminant analysis. In *The European Conference on Machine Learning and Principles and Practice of Knowledge Discovery in Databases (ECML PKDD)*, 2009.

[15] Z. Zhang and M. I. Jordan. Multiway spectral clustering: A margin-based perspective. *Statistical Science*, 23(3):383–403, 2008.

[16] H. Zou and T. Hastie. Regularization and variable selection via the elastic net. *Journal of the Royal Statistical Society, Series B*, 67:301–320, 2005.

[17] H. Zou, T. Hastie, and R. Tibshirani. Sparse principal component analysis. *Journal of Computational and Graphical Statistics*, 15:265–286, 2006.

